# Analysis of Representations for Domain Adaptation

**Shai Ben-David**
School of Computer Science
University of Waterloo
shai@cs.uwaterloo.ca

**John Blitzer, Koby Crammer, and Fernando Pereira**
Department of Computer and Information Science
University of Pennsylvania
{blitzer, crammer, pereira}@cis.upenn.edu

## Abstract

Discriminative learning methods for classification perform well when training and test data are drawn from the same distribution. In many situations, though, we have labeled training data for a *source* domain, and we wish to learn a classifier which performs well on a *target* domain with a different distribution. Under what conditions can we adapt a classifier trained on the source domain for use in the target domain? Intuitively, a good feature representation is a crucial factor in the success of domain adaptation. We formalize this intuition theoretically with a generalization bound for domain adaption. Our theory illustrates the tradeoffs inherent in designing a representation for domain adaptation and gives a new justification for a recently proposed model. It also points toward a promising new model for domain adaptation: one which explicitly minimizes the difference between the source and target domains, while at the same time maximizing the margin of the training set.

## 1 Introduction

We are all familiar with the situation in which someone learns to perform a task on training examples drawn from some domain (the *source* domain), but then needs to perform the same task on a related domain (the *target domain*). In this situation, we expect the task performance in the target domain to depend on both the performance in the source domain and the similarity between the two domains.

This situation arises often in machine learning. For example, we might want to adapt for a new user (the target domain) a spam filter trained on the email of a group of previous users (the source domain), under the assumption that users generally agree on what is spam and what is not. Then, the challenge is that the distributions of emails for the first set of users and for the new user are different. Intuitively, one might expect that the closer the two distributions are, the better the filter trained on the source domain will do on the target domain.

Many other instances of this situation arise in natural language processing. In general, labeled data for tasks like part-of-speech tagging, parsing, or information extraction are drawn from a limited set of document types and genres in a given language because of availability, cost, and project goals. However, applications for the trained systems often involve somewhat different document types and genres. Nevertheless, part-of-speech, syntactic structure, or entity mention decisions are to a large extent stable across different types and genres since they depend on general properties of the language under consideration.

Discriminative learning methods for classification are based on the assumption that training and test data are drawn from the same distribution. This assumption underlies both theoretical estimates of generalization error and the many experimental evaluations of learning methods. However, the assumption does not hold for domain adaptation [5, 7, 13, 6]. For the situations we outlined above, the challenge is the difference in instance distribution between the source and target domains. We will approach this challenge by investigating how a common representation between the two domains

can make the two domains appear to have similar distributions, enabling effective domain adaptation. We formalize this intuition with a bound on the *target* generalization error of a classifier trained from labeled data in the *source* domain. The bound is stated in terms of a representation function, and it shows that a representation function should be designed to minimize domain divergence, as well as classifier error.

While many authors have analyzed adaptation from multiple sets of labeled training data [3, 5, 7, 13], our theory applies to the setting in which the target domain has no labeled training data, but plentiful unlabeled data exists for both target and source domains. As we suggested above, this setting realistically captures the problems widely encountered in real-world applications of machine learning. Indeed recent empirical work in natural language processing [11, 6] has been targeted at exactly this setting.

We show experimentally that the heuristic choices made by the recently proposed structural correspondence learning algorithm [6] do lead to lower values of the relevant quantities in our theoretical analysis, providing insight as to why this algorithm achieves its empirical success. Our theory also points to an interesting new algorithm for domain adaptation: one which directly minimizes a trade-off between source-target similarity and source training error.

The remainder of this paper is structured as follows: In the next section we formally define domain adaptation. Section 3 gives our main theoretical results. We discuss how to compute the bound in section 4. Section 5 shows how the bound behaves for the structural correspondence learning representation [6] on natural language data. We discuss our findings, including a new algorithm for domain adaptation based on our theory, in section 6 and conclude in section 7.

## 2   Background and Problem Setup

Let $\mathcal{X}$ be an instance set. In the case of [6], this could be all English words, together with the possible contexts in which they occur. Let $\mathcal{Z}$ be a feature space ($\mathbb{R}^d$ is a typical choice) and $\{0, 1\}$ be the label set for binary classification[1].

A learning problem is specified by two parameters: a distribution $\mathcal{D}$ over $\mathcal{X}$ and a (stochastic) target function $f : \mathcal{X} \rightarrow [0, 1]$. The value of $f(\mathbf{x})$ corresponds to the probability that the label of $\mathbf{x}$ is 1. A representation function $\mathcal{R}$ is a function which maps instances to features $\mathcal{R} : \mathcal{X} \rightarrow \mathcal{Z}$. A representation $\mathcal{R}$ induces a distribution over $\mathcal{Z}$ and a (stochastic) target function from $\mathcal{Z}$ to $[0, 1]$ as follows:

$$\Pr_{\tilde{\mathcal{D}}} [B] \quad \overset{\text{def}}{=} \quad \Pr_{\mathcal{D}} \left[ \mathcal{R}^{-1}(B) \right]$$

$$\tilde{f}(\mathbf{z}) \quad \overset{\text{def}}{=} \quad \mathrm{E}_{\mathcal{D}} \left[ f(\mathbf{x}) | \mathcal{R}(\mathbf{x}) = \mathbf{z} \right]$$

for any $A \subseteq Z$ such that $\mathcal{R}^{-1}(B)$ is $\mathcal{D}$-measurable. In words, the probability of an event $B$ under $\tilde{\mathcal{D}}$ is the probability of the inverse image of $B$ under $\mathcal{R}$ according to $\mathcal{D}$, and the probability that the label of $\mathbf{z}$ is 1 according to $\tilde{f}$ is the mean of probabilities of instances $\mathbf{x}$ that $\mathbf{z}$ represents. Note that $\tilde{f}(\mathbf{z})$ may be a stochastic function even if $f(\mathbf{x})$ is not. This is because the function $\mathcal{R}$ can map two instances with different $f$-labels to the same feature representation. In summary, our learning setting is defined by fixed but unknown $\mathcal{D}$ and $f$, and our choice of representation function $\mathcal{R}$ and hypothesis class $\mathcal{H} \subseteq \{g : \mathcal{Z} \rightarrow \{0, 1\}\}$ of deterministic hypotheses to be used to approximate the function $f$.

### 2.1   Domain Adaptation

We now formalize the problem of *domain adaptation*. A *domain* is a distribution $\mathcal{D}$ on the instance set $\mathcal{X}$. Note that this is *not* the domain of a function. To avoid confusion, we will always mean a specific distribution over the instance set when we say domain. Unlike in inductive transfer, where the tasks we wish to perform may be related but different, in domain adaptation we perform the *same* task in multiple domains. This is quite common in natural language processing, where we might be performing the same syntactic analysis task, such as tagging or parsing, but on domains with very different vocabularies [6, 11].

We assume two domains, a *source* domain and a *target* domain. We denote by $\mathcal{D}_S$ the source distribution of instances and $\tilde{\mathcal{D}}_S$ the induced distribution over the feature space $\mathcal{Z}$. We use parallel notation, $\mathcal{D}_T$, $\tilde{\mathcal{D}}_T$, for the target domain. $f : \mathcal{X} \rightarrow [0,1]$ is the labeling rule, common to both domains, and $\tilde{f}$ is the induced image of $f$ under $\mathcal{R}$.

A predictor is a function, $h$, from the feature space, $\mathcal{Z}$ to $[0,1]$. We denote the probability, according the distribution $\mathcal{D}_S$, that a predictor $h$ disagrees with $f$ by

$$
\begin{aligned}
\epsilon_S(h) &= \mathrm{E}_{\mathbf{z}\sim\tilde{\mathcal{D}}_S}\left[\mathrm{E}_{y\sim\tilde{f}(\mathbf{z})}\left[y \neq h(\mathbf{z})\right]\right] \\
&= \mathrm{E}_{\mathbf{z}\sim\tilde{\mathcal{D}}_S}\left|\tilde{f}(\mathbf{z}) - h(\mathbf{z})\right| .
\end{aligned}
$$

Similarly, $\epsilon_T(h)$ denotes the expected error of $h$ with respect to $\mathcal{D}_T$.

# 3 Generalization Bounds for Domain Adaptation

We now proceed to develop a bound on the target domain generalization performance of a classifier trained in the source domain. As we alluded to in section 1, the bound consists of two terms. The first term bounds the performance of the classifier on the *source* domain. The second term is a measure of the divergence between the induced source marginal $\tilde{\mathcal{D}}_S$ and the induced target marginal $\tilde{\mathcal{D}}_T$. A natural measure of divergence for distributions is the $L_1$ or variational distance. This is defined as

$$
d_{L_1}(\mathcal{D},\mathcal{D}') = 2\sup_{B\in\mathcal{B}}|\mathrm{Pr}_{\mathcal{D}}\left[B\right] - \mathrm{Pr}_{\mathcal{D}'}\left[B\right]|
$$

where $\mathcal{B}$ is the set of measureable subsets under $\mathcal{D}$ and $\mathcal{D}'$. Unfortunately the variational distance between real-valued distributions cannot be computed from finite samples [2, 9] and therefore is not useful to us when investigating representations for domain adaptation on real-world data.

A key part of our theory is the observation that in many realistic domain adaptation scenarios, we do not need such a powerful measure as variational distance. Instead we can restrict our notion of domain distance to be measured only with respect to function in our hypothesis class.

## 3.1 The $\mathcal{A}$-distance and labeling function complexity

We make use of a special measure of distance between probability distributions, the $\mathcal{A}$-distance, as introduced in [9]. Given a domain $\mathcal{X}$ and a collection $\mathcal{A}$ of subsets of $\mathcal{X}$, let $\mathcal{D}$, $\mathcal{D}'$ be probability distributions over $\mathcal{X}$, such that every set in $\mathcal{A}$ is measurable with respect to both distributions. the $\mathcal{A}$-distance between such distributions is defined as

$$
d_{\mathcal{A}}(\mathcal{D},\mathcal{D}') = 2\sup_{A\in\mathcal{A}}|\mathrm{Pr}_{\mathcal{D}}\left[A\right] - \mathrm{Pr}_{\mathcal{D}'}\left[A\right]|
$$

In order to use the $\mathcal{A}$-distance, we need to limit the complexity of the true function $f$ in terms of our hypothesis class $\mathcal{H}$. We say that a function $\tilde{f} : \mathcal{Z} \rightarrow [0,1]$ is $\lambda$-close to a function class $\mathcal{H}$ with respect to distributions $\tilde{\mathcal{D}}_S$ and $\tilde{\mathcal{D}}_T$ if

$$
\inf_{h\in\mathcal{H}}\left[\epsilon_S(h) + \epsilon_T(h)\right] \leq \lambda .
$$

A function $\tilde{f}$ is $\lambda$-close to $\mathcal{H}$ when there is a single hypothesis $h \in \mathcal{H}$ which performs well on *both* domains. This embodies our domain adaptation assumption, and we will assume will assume that our induced labeling function $\tilde{f}$ is $\lambda$-close to our hypothesis class $\mathcal{H}$ for a small $\lambda$.

We briefly note that in standard learning theory, it is possible to achieve bounds with no explicit assumption on labeling function complexity. If $\mathcal{H}$ has bounded capacity (e.g., a finite VC-dimension), then uniform convergence theory tells us that whenever $\tilde{f}$ is not $\lambda$-close to $\mathcal{H}$, large training samples have poor empirical error for every $h \in H$. This is *not* the case for domain adaptation. If the training data is generated by some $D_S$ and we wish to use some $\mathcal{H}$ as a family of predictors for labels in the target domain, $T$, then one can construct a function which agrees with some $h \in \mathcal{H}$ with respect to $\tilde{\mathcal{D}}_S$ and yet is far from $\mathcal{H}$ with respect to $\tilde{\mathcal{D}}_T$. Nonetheless we believe that such examples do not occur for realistic domain adaptation problems when the hypothesis class $\mathcal{H}$ is sufficiently rich, since for most domain adaptation problems of interest the labeling function is 'similarly simple' for both the source and target domains.

## 3.2 Bound on the target domain error

We require one last piece of notation before we state and prove the main theorems of this work: the correspondence between functions and characteristic subsets. For a binary-valued function $g(\mathbf{z})$, we let $\mathcal{Z}_g \subseteq \mathcal{Z}$ be the subset whose characteristic function is $g$

$$\mathcal{Z}_g = \{\mathbf{z} \in \mathcal{Z} : g(\mathbf{z}) = 1\} \ .$$

In a slight abuse of notation, for a binary function class $\mathcal{H}$ we will write $d_\mathcal{H}(\cdot, \cdot)$ to indicate the $\mathcal{A}$-distance on the class of subsets whose characteristic functions are functions in $\mathcal{H}$. Now we can state our main theoretical result.

**Theorem 1** *Let $\mathcal{R}$ be a fixed representation function from $\mathcal{X}$ to $\mathcal{Z}$ and $\mathcal{H}$ be a hypothesis space of VC-dimension $d$. If a random labeled sample of size $m$ is generated by applying $R$ to a $\mathcal{D}_S$-i.i.d. sample labeled according to $f$, then with probability at least $1 - \delta$, for every $h \in \mathcal{H}$:*

$$\epsilon_T(h) \leq \hat{\epsilon}_S(h) + \sqrt{\frac{4}{m}\left(d\log\frac{2em}{d} + \log\frac{4}{\delta}\right)} + d_\mathcal{H}(\tilde{\mathcal{D}}_S, \tilde{\mathcal{D}}_T) + \lambda$$

*where $e$ is the base of the natural logarithm.*

**Proof:** Let $h^* = \operatorname{argmin}_{h \in H}\left(\epsilon_T(h) + \epsilon_S(h)\right)$, and let $\lambda_T$ and $\lambda_S$ be the errors of $h^*$ with respect to $\mathcal{D}_T$ and $\mathcal{D}_S$ respectively. Notice that $\lambda = \lambda_T + \lambda_S$.

$$
\begin{aligned}
\epsilon_T(h) &\leq \lambda_T + \Pr_{\mathcal{D}_T}\left[\mathcal{Z}_h \Delta \mathcal{Z}_{h^*}\right] \\
&\leq \lambda_T + \Pr_{\mathcal{D}_S}\left[\mathcal{Z}_h \Delta \mathcal{Z}_{h^*}\right] + \left|\Pr_{\mathcal{D}_S}\left[\mathcal{Z}_h \Delta \mathcal{Z}_{h^*}\right] - \Pr_{\mathcal{D}_T}\left[\mathcal{Z}_h \Delta \mathcal{Z}_{h^*}\right]\right| \\
&\leq \lambda_T + \Pr_{\mathcal{D}_S}\left[\mathcal{Z}_h \Delta \mathcal{Z}_{h^*}\right] + d_\mathcal{H}(\tilde{\mathcal{D}}_S, \tilde{\mathcal{D}}_T) \\
&\leq \lambda_T + \lambda_S + \epsilon_S(h) + d_\mathcal{H}(\tilde{\mathcal{D}}_S, \tilde{\mathcal{D}}_T) \\
&\leq \lambda + \epsilon_S(h) + d_\mathcal{H}(\tilde{\mathcal{D}}_S, \tilde{\mathcal{D}}_T)
\end{aligned}
$$

The theorem now follows by a standard application Vapnik-Chervonenkis theory [14] to bound the true $\epsilon_S(h)$ by its empirical estimate $\hat{\epsilon}_S(h)$. Namely, if $S$ is an $m$-size .i.i.d. sample, then with probability exceeding $1 - \delta$,

$$\epsilon_S(h) \leq \hat{\epsilon}_S(h) + \sqrt{\frac{4}{m}\left(d\log\frac{2em}{d} + \log\frac{4}{\delta}\right)}$$

∎ The bound depends on the quantity $d_\mathcal{H}(\tilde{\mathcal{D}}_S, \tilde{\mathcal{D}}_T)$. We chose the $\mathcal{A}$-distance, however, precisely because we can measure this from finite samples from the distrbutions $\tilde{\mathcal{D}}_S$ and $\tilde{\mathcal{D}}_T$ [9]. Combining 1 with theorem 3.2 from [9], we can state a computable bound for the error on the target domain.

**Theorem 2** *Let $\mathcal{R}$ be a fixed representation function from $\mathcal{X}$ to $\mathcal{Z}$ and $\mathcal{H}$ be a hypothesis space of VC-dimension $d$.*

*If a random labeled sample of size $m$ is generated by applying $R$ to a $\mathcal{D}_S$ - i.i.d. sample labeled according to $f$, and $\tilde{\mathcal{U}}_S$, $\tilde{\mathcal{U}}_T$ are unlabeled samples of size $m'$ each, drawn from $\tilde{\mathcal{D}}_S$ and $\tilde{\mathcal{D}}_T$ respectively, then with probability at least $1 - \delta$ (over the choice of the samples), for every $h \in \mathcal{H}$:*

$$\epsilon_T(h) \leq \hat{\epsilon}_S(h) + \frac{4}{m}\sqrt{\left(d\log\frac{2em}{d} + \log\frac{4}{\delta}\right)} + \lambda + d_\mathcal{H}(\tilde{\mathcal{U}}_S, \tilde{\mathcal{U}}_T) + 4\sqrt{\frac{d\log(2m') + \log(\frac{4}{\delta})}{m'}}$$

Let us briefly examine the bound from theorem 2, with an eye toward feature representations, $\mathcal{R}$. Under the assumption of subsection 3.1, we assume that $\lambda$ is small for reasonable $\mathcal{R}$. Thus the two main terms of interest are the first and fourth terms, since the representation $\mathcal{R}$ directly affects them. The first term is the empirical training error. The fourth term is the sample $\mathcal{A}$-distance between domains for hypothesis class $\mathcal{H}$. Looking at the two terms, we see that a good representation $\mathcal{R}$ is one which achieves low values for both training error and domain $\mathcal{A}$-distance simultaneously.

# 4 Computing the $\mathcal{A}$-distance for Signed Linear Classifiers

In this section we discuss practical considerations in computing the $\mathcal{A}$-distance on real data. Ben-David et al. [9] show that the $\mathcal{A}$-distance can be approximated arbitrarily well with increasing sample size. Recalling the relationship between sets and their characteristic functions, it should be clear that computing the $A$-distance is closely related to learning a classifier. In fact they are identical. The set $A_h \in \mathcal{H}$ which maximizes the $\mathcal{H}$-distance between $\tilde{\mathcal{D}}_S$ and $\tilde{\mathcal{D}}_T$ has a characteristic function $h$. Then $h$ is the classifier which achieves minimum error on the binary classification problem of discriminating between points generated by the two distributions.

To see this, suppose we have two samples $\tilde{\mathcal{U}}_S$ and $\tilde{\mathcal{U}}_T$ , each of size $m'$ from $\tilde{\mathcal{D}}_S$ and $\tilde{\mathcal{D}}_T$ respectively. Define the error of a classifier $h$ on the task of discriminating between points sampled from different distributions as

$$\mathrm{err}(\mathrm{h}) = \frac{1}{2\mathrm{m}'} \sum_{\mathrm{i}=1}^{2\mathrm{m}'} \left| \mathrm{h}(\mathbf{z}_\mathrm{i}) - \mathrm{I}_{\mathbf{z}_\mathrm{i} \in \tilde{\mathcal{U}}_\mathrm{S}} \right| \ ,$$

where $I_{\mathbf{z}_i \in \tilde{\mathcal{U}}_S}$ is the indicator function for points lying in the sample $\tilde{\mathcal{U}}_S$. In this case, it is straightforward to show that

$$d_A(\tilde{\mathcal{U}}_S, \tilde{\mathcal{U}}_T) = 2 \left( 1 - 2 \min_{h' \in \mathcal{H}} \mathrm{err}(\mathrm{h}') \right) \ .$$

Unfortunately it is a known NP-hard problem even to approximate the error of the optimal hyperplane classifier for arbitrary distributions [4]. We choose to approximate the optimal hyperplane classifier by minimizing a convex upper bound on the error, as is standard in classification. It is important to note that this does *not* provide us with a valid upper bound on the target error, but as we will see it nonetheless provides us with useful insights about representations for domain adaptation. In the subsequent experiments section, we train a linear classifier to discriminate between points sampled from different domains to illustrate a proxy for the $\mathcal{A}$-distance. We minimize a modified Huber loss using stochastic gradient descent, described more completely in [15].

# 5 Natural Language Experiments

In this section we use our theory to analyze different representations for the task of adapting a part of speech tagger from the financial to biomedical domains [6]. The experiments illustrate the utility of the bound and all of them have the same flavor. First, we choose a representation $\mathcal{R}$. Then we train a classifier using $\mathcal{R}$ and measure the different terms of the bound. As we shall see, represenations which minimize both relevant terms of the bound also have small empirical error.

Part of speech (PoS) tagging is the task of labeling a word in context with its grammatical function. For instance, in the previous sentence we would the tag for "speech" is *singular common noun*, the tag for "labeling" is *gerund*, and so on. PoS tagging is a common preprocessing step in many pipelined natural language processing systems and is described in more detail in [6]. Blitzer et al. empirically investigate methods for adpating a part of speech tagger from financial news (the Wall Street Journal, henceforth also WSJ) to biomedical abstracts (MEDLINE) [6]. We have obtained their data, and we will use it throughout this section. As in their investigation, we treat the financial data as our source, for which we have labeled training data and the biomedical abstracts as our target, for which we have no labeled training data.

The representations we consider in this section are all linear projections of the original feature space into $\mathbb{R}^d$. For PoS tagging, the original feature space consists of high-dimensional, sparse binary vectors [6]. In all of our experiments we choose $d$ to be 200. Now at train time we apply the projection to the binary feature vector representation of each instance and learn a linear classifier in the $d$-dimensional projected space. At test time we apply the projection to the binary feature vector representation and classify in the $d$-dimensional projected space.

## 5.1 Random Projections

If our original feature space is of dimension $d'$, our random projection matrix is a matrix $P \in \mathbb{R}^{d \times d'}$. The entries of $P$ are drawn i.i.d. from $\mathcal{N}(0, 1)$. The Johnson-Lindenstrauss lemma [8] guarantees

**(a)** Plot of SCL representation for financial (squares) vs. biomedical (circles)  **(b)** Plot of SCL representation for nouns (diamonds) vs. verbs (triangles)

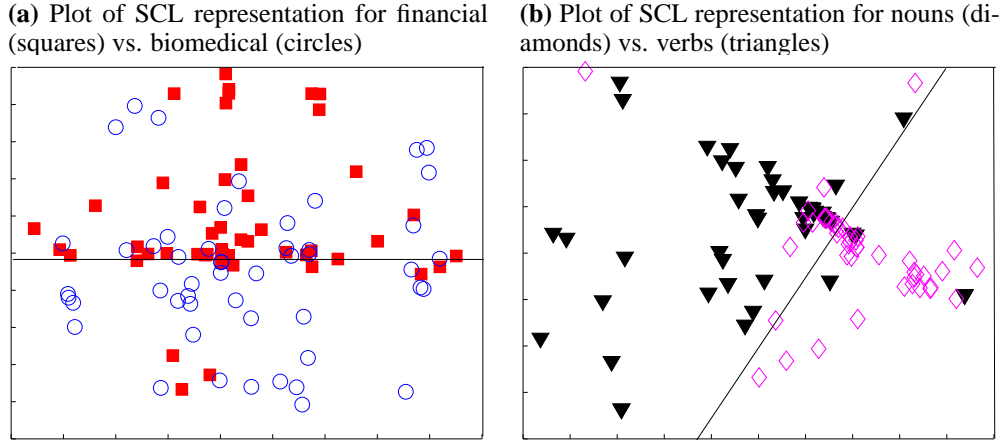

Figure 1: 2D plots of SCL representations for the **(a)** $\mathcal{A}$-distance and **(b)** empirical risk parts of theorem 2

that random projections approximate well distances in the original high dimensional space, as long as $d$ is sufficiently large. Arriaga and Vempala [1] show that one can achieve good prediction with random projections as long as the margin is sufficiently large.

## 5.2 Structural Correspondence Learning

Blitzer et al. [6] describe a heuristic method for domain adaptation that they call structural correspondence learning (henceforth also SCL). SCL uses unlabeled data from both domains to induce correspondences among features in the two domains. Its first step is to identify a small set of domain-independent "pivot" features which occur frequently in the unlabeled data of both domains. Other features are then represented using their relative co-occurrence counts with these pivot features. Finally they use a low-rank approximation to the co-occurence count matrix as a projection matrix $P$. The intuition is that by capturing these important correlations, features from the source and target domains which behave similarly for PoS tagging will be represented similarly in the projected space.

## 5.3 Results

We use as our *source* data set 100 sentences (about 2500 words) of PoS-tagged Wall Street Journal text. The *target* domain test set is the same set as in [6]. We use one million words (500 thousand from each domain) of unlabeled data to estimate the $\mathcal{A}$-distance between the financial and biomedical domains.

The results in this section are intended to illustrate the different parts of theorem 2 and how they can affect the target domain generalization error. We give two types of results. The first are pictorial and appear in figures 1(a), 1(b) and 2(a). These are intended to illustrate either the $\mathcal{A}$-distance (figures 1(a) and 2(a)) or the empirical error (figure 1(b)) for different representations. The second type are empirical and appear in 2(b). In this case we use the Huber loss as a proxy from the empirical training error.

Figure 1(a) shows one hundred random instances projected onto the space spanned by the best two discriminating projections from the SCL projection matrix for part of the financial and biomedical dataset. Instances from the WSJ are depicted as filled red squares, whereas those from MEDLINE are depicted as empty blue circles. An approximating linear discrimnator is also shown. Note, however, that the discriminator performs poorly, and recall that if the best discriminator performs poorly the $\mathcal{A}$-distance is low. On the other hand, figure 1(b) shows the best two discriminating components for the task of discriminating between nouns and verbs. Note that in this case, a good discriminating divider is easy to find, even in such a low-dimensional space. Thus these pictures lead us to believe that SCL finds a representation which results both in small empirical classification error and small $\mathcal{A}$-distance. In this case theorem 2 predicts good performance.

**(a)** Plot of random projections representation for financial (squares) vs. biomedical (circles)

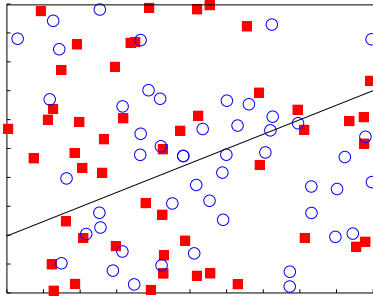

**(b)** Comparison of bound terms vs.target domain error for different choices of representation. **Reprentations** are linear projections of the original feature space. **Huber loss** is the *labeled* training loss after training, and the $\mathcal{A}$-**distance** is approximated as described in the previous subsection. **Error** refers to tagging error for the full tagset on the target domain.

| **Representation** | Huber loss | $\mathcal{A}$-distance | Error |
|---|---|---|---|
| Identity | **0.003** | 1.796 | 0.253 |
| Random Proj | 0.254 | 0.223 | 0.561 |
| SCL | 0.07 | **0.211** | **0.216** |

Figure 2: **(a)** 2D plot of random projection representation and **(b)** results summary on large data

Figure 2(a) shows one hundred random instances projected onto the best two discriminating projections for WSJ vs. MEDLINE from a random matrix of 200 projections. This also seems to be difficult to separate. The random projections don't reveal any useful structure for learning, either, though. Not shown is the corresponding noun vs. verb plot for random projections. It looks identical to 2(a). Thus theorem 2 predicts that using two random projections as a representation will perform poorly, since it minimizes only the $\mathcal{A}$-distance and not the empirical error.

Figure 2(b) gives results on a large training and test set showing how the value of the bound can affect results. The identity representation achieves very low Huber loss (corresponding to empirical error). The original feature set consists of 3 million binary-valued features, though, and it is quite easy to separate the two domains using these features. The approximate $\mathcal{A}$-distance is near the maximum possible value.

The random projections method achieves low $\mathcal{A}$-distance but high Huber loss, and the classifier which uses this representation achieves error rates much lower than the a classifier which uses the identity representation. Finally, the structural correspondence learning representation achieves low Huber loss and low $\mathcal{A}$-distance, and the error rate is the lowest of the three representations.

## 6   Discussion and Future Work

Our theory demonstrates an important tradeoff inherent in designing good representations for domain adaptation. A good representation enables achieving low error rate on the source domain while also minimizing the $\mathcal{A}$-distance between the induced marginal distributions of the two domains. The previous section demonstrates empirically that the heuristic choices of the SCL algorithm [6] do achieve low values for each of these terms.

Our theory is closely related to theory by Sugiyama and Mueller on covariate shift in regression models [12]. Like this work, they consider the case where the prediction functions are identical, but the input data (covariates) have different distributions. Unlike their work, though, we bound the target domain error using a finite source domain labeled sample and finite source and target domain unlabeled samples.

Our experiments illustrate the utility of our bound on target domain error, but they do not explore the accuracy of our approximate $\mathcal{H}$-distance. This is an important area of exploration for future work. Finally our theory points toward an interesting new direction for domain adapation. Rather than heuristically choosing a representation, as previous research has done [6], we can try to learn a representation which directly minimizes a combination of the terms in theorem 2. If we learn mappings from some parametric family (linear projections, for example), we can give a bound on the error in terms of the complexity of this family. This may do better than the current heuristics, and we are also investigating theory and algorithms for this.

# 7 Conclusions

We presented an analysis of representations for domain adaptation. It is reasonable to think that a good representation is the key to effective domain adaptation, and our theory backs up that intuition. Theorem 2 gives an upper bound on the generalization of a classifier trained on a *source* domain and applied in a *target* domain. The bound depends on the representation and explicitly demonstrates the tradeoff between low empirical source domain error and a small difference between distributions.

Under the assumption that the labeling function $\tilde{f}$ is close to our hypothesis class $\mathcal{H}$, we can compute the bound from finite samples. The relevant distributional divergence term can be written as the $\mathcal{A}$-distance of Kifer *et al* [9]. Computing the $\mathcal{A}$-distance is equivalent to finding the minimum-error classifier. For hyperplane classifiers in $\mathbb{R}^d$, this is an NP-hard problem, but we give experimental evidence that minimizing a convex upper bound on the error, as in normal classification, can give a reasonable approximation to the $\mathcal{A}$-distance.

Our experiments indicate that the heuristic structural correspondence learning method [6] does in fact simultaneously achieve low $\mathcal{A}$-distance as well as a low margin-based loss. This provides a justification for the heuristic choices of SCL "pivots". Finally we note that our theory points to an interesting new algorithm for domain adaptation. Instead of making heuristic choices, we are investigating algorithms which directly minimize a combination of the $\mathcal{A}$-distance and the empirical training margin.

## Footnotes

[1]The same type of analysis hold for multiclass classification, but for simplicty we analyze the binary case.

# References

[1] R. Arriaga and S. Vempala. An algorithmic theory of learning robust concepts and random projection. In *FOCS*, volume 40, 1999.

[2] T. Batu, L. Fortnow, R. Rubinfeld, W. Smith, and P. White. Testing that distributions are close. In *FOCS*, volume 41, pages 259–269, 2000.

[3] J. Baxter. Learning internal representations. In *COLT '95: Proceedings of the eighth annual conference on Computational learning theory*, pages 311–320, New York, NY, USA, 1995.

[4] S. Ben-David, N. Eiron, and P. Long. On the difficulty of approximately maximizing agreements. *Journal of Computer and System Sciences*, 66:496–514, 2003.

[5] S. Ben-David and R. Schuller. Exploiting task relatedness for multiple task learning. In *COLT 2003: Proceedings of the sixteenth annual conference on Computational learning theory*, 2003.

[6] J. Blitzer, R. McDonald, and F. Pereira. Domain adaption with structural correspondence learning. In *EMNLP*, 2006.

[7] K. Crammer, M. Kearns, and J. Wortman. Learning from data of variable quality. In *Neural Information Processing Systems (NIPS)*, Vancouver, Canada, 2005.

[8] W. Johnson and J. Lindenstrauss. Extension of lipschitz mappings to hilbert space. *Contemporary Mathematics*, 26:189–206, 1984.

[9] D. Kifer, S. Ben-David, and J. Gehrke. Detecting change in data streams. In *Very Large Databases (VLDB)*, 2004.

[10] C. Manning. *Foundations of Statistical Natural Language Processing*. MIT Press, Boston, 1999.

[11] D. McClosky, E. Charniak, and M. Johnson. Reranking and self-training for parser adaptation. In *ACL*, 2006.

[12] M. Sugiyama and K. Mueller. Generalization error estimation under covariate shift. In *Workshop on Information-Based Induction Sciences*, 2005.

[13] Y. W. Teh, M. I. Jordan, M. J. Beal, and D. M. Blei. Sharing clusters among related groups: Hierarchical Dirichlet processes. In *Advances in Neural Information Processing Systems*, volume 17, 2005.

[14] V. Vapnik. *Statistical Learning Theory*. John Wiley, New York, 1998.

[15] T. Zhang. Solving large-scale linear prediction problems with stochastic gradient descent. In *ICML*, 2004.
